# Dynamic Cell Structures

**Jörg Bruske and Gerald Sommer**

Department of Cognitive Systems
Christian Albrechts University at Kiel
24105 Kiel - Germany

## Abstract

**Dynamic Cell Structures** (DCS) represent a family of artificial neural architectures suited both for **unsupervised** and **supervised** learning. They belong to the recently [Martinetz94] introduced class of **Topology Representing Networks** (TRN) which build **perfectly topology preserving feature maps**. DCS employ a modified **Kohonen learning rule** in conjunction with **competitive Hebbian learning**. The Kohonen type learning rule serves to adjust the synaptic weight vectors while Hebbian learning establishes a dynamic **lateral connection structure** between the units reflecting the topology of the feature manifold. In case of supervised learning, i.e. function approximation, each neural unit implements a **Radial Basis Function**, and an additional layer of linear output units adjusts according to a **delta-rule**. DCS is the first RBF-based approximation scheme attempting to concurrently learn and utilize a perfectly topology preserving map for improved performance.
Simulations on a selection of CMU-Benchmarks indicate that the DCS idea applied to the **Growing Cell Structure** algorithm [Fritzke93] leads to an efficient and elegant algorithm that can beat conventional models on similar tasks.

## 1 Introduction

The quest for smallest topology preserving maps motivated the introduction of growing feature maps like Fritzke's Growing Cell Structures (GCS). In **GCS**, see [Fritzke93] for details, one starts with a $k$-dimensional simplex of $N = k+1$ neural units and $(k + 1) \cdot k/2$ lateral connections (edges). Growing of the network is performed such that after insertion

of a new unit the network consists solely of $k$ dimensional simplices again. Thus, like Kohonen's SOM, **GCS** can only learn a **perfectly topology preserving feature map**[1] if $k$ meets the actual dimension of the feature manifold. Assuming that the lateral connections do reflect the actual topology the connections serve to define a neighborhood for a Kohonen like adaptation of the synaptic vectors $w_i$ and guide the insertion of new units. Insertion happens incrementally and does not necessitate a retraining of the network. The principle is to insert new neurons in such a way that the expected value of a certain local error measure, which Fritzke calls the **resource**, becomes equal for all neurons. For instance, the number of times a neuron wins the competition, the sum of distances to stimuli for which the neuron wins or the sum of errors in the neuron's output can all serve as a resource and dramatically change the behavior of **GCS**. Using different error measures and guiding insertion by the lateral connections contributes much to the success of **GCS**.

The principle of DCS is to avoid any restriction of the topology of the network (lateral connection scheme between the neural units) but to concurrently learn and utilize a **perfectly topology preserving map**. This is achieved by adapting the lateral connection structure according to a **competitive Hebbian learning rule**[2]:

$$C_{ij}(t+1) = \begin{cases} max\{y_i \cdot y_j, C_{ij}(t)\} & : \ y_i \cdot y_j \geq y_k \cdot y_l \ \forall \, (1 \leq k, l \leq N) \\ 0 & : \ C_{ij}(t) < \theta \quad \quad "" \\ \alpha C_{ij}(t) & : \ \text{otherwise}, \quad \quad "" \end{cases} \quad (1)$$

where $\alpha, 0 < \alpha < 1$ is a forgetting constant, $\theta, 0 < \theta < 1$ serves as a threshold for deleting lateral connections, and $y_i = R(\|v - w_i\|)$ is the activation of the i-th unit with $w_i$ as the centre of its receptive field on presentation of stimulus $v$. $R(.)$ can be any positive continuously monotonically decreasing function. For batch learning with a training set $T$ of fixed size $|T|$, $\alpha = \sqrt[|T|]{\theta}$ is a good choice.

Since the isomorphic representation of the topology of the feature manifold $M$ in the lateral connection structure is central to performance, in many situations a **DCS** algorithm may be the right choice. These situations are characterized by missing **a priori** knowledge of the topology of the feature manifold $M$ or a topology of $M$ which cannot be readily mapped to the existing models. Of course, if such a priori knowledge is available then models like **GCS** or Kohonen's **SOM** allowing to incorporate such knowledge have an advantage, especially if training data are sparse.

Note that **DCS** algorithms can also aid in cluster analysis: In a perfectly topology preserving map clusters which are bounded by regions of $P(v) = 0$ can be identified simply by a connected component analysis. However, without prior knowledge about the feature manifold $M$ it is in principal impossible to check for perfect topology preservation of $S$. Noise in the input data may render perfect topology learning even more difficult. So what can perfect topology learning be used for? The answer is simply that for every set $S$ of reference vectors perfect topology learning yields maximum topology preservation[3] with respect to this set. And connected components with respect to the lateral connection structure $C$ may well serve as an initialization for postprocessing by hierarchical cluster algorithms.

---

1. We use the term "perfectly topology preserving feature map" in accordance with its rigorous definition in [Martinetz93].

2. In his very recent and recommendable article [Martinetz94] the term Topology Representing Network (TRN) is coined for any network employing competitive Hebbian learning for topology learning.

3. if topology preservation is measured by the topographic function as defined in [Villmann94].

The first neural algorithm attempting to learn **perfectly topology preserving feature maps** is the **Neural Gas** algorithm of T. Martinetz [Martinetz92]. However, unlike **DCS** the **Neural Gas** does not further exploit this information: In every step the **Neural Gas** computes the k nearest neighbors to a given stimulus and, in the supervised case, employs all of them for function approximation. **DCS** avoids this computational burden by utilizing the lateral connection structure (topology) learned so far, and it restricts interpolation between activated units to the submanifold of the current stimulus.

Applying the principle of **DCS** to Fritzke's **GCS** yields our **DCS-GCS** algorithm. This algorithm sticks very closely to the basic structure of its ancestor **GCS** except the predefined $k$-dimensional simplex connection structure being replaced by perfect topology learning. Besides the conceptual advantage of perfect topology learning, **DCS-GCS** does decrease overhead (Fritzke has to handle quite sophisticated data structures in order to maintain the $k$-dimensional simplex structure after insertion/ deletion of units) and can be readily implemented on any serial computer.

## 2    Unsupervised DCS-GCS

The unsupervised DCS-GCS algorithm starts with initializing the network (graph) to two neural units (vertices) $n_1$ and $n_2$. Their weight vectors $w_1$, $w_1$ (centres of receptive fields) are set to points $v_1, v_2 \in M$ which are drawn from $M$ according to P(v). They are connected by a lateral connection of weight $C_{12} = C_{21} = 1$. Note that lateral connections in **DCS** are always bidirectional and have symmetric weights.

Now the algorithm enters its outer loop which is repeated until some stopping criterium is fulfilled. This stopping criterium could for instance be a test whether the quantization error has already dropped below a predefined accuracy.

The inner loop is repeated for $\lambda$ times. In off-line learning $\lambda$ can be set to the number examples in the training set $T$. In this case, the inner loop just represents an epoch of training.

Within the inner loop, the algorithm first draws an input stimulus $v \in M$ from $M$ according to P(v) and then proceeds to calculate the two neural units which weight vectors are first and second closest to $v$.

In the next step, the lateral connections between the neural units are modified according to eq. (1), the **competitive Hebbian learning rule**. As has already been mentioned, in off-line learning it is a good idea to set $\alpha = \sqrt[\lambda]{\theta}$.

Now the weight vectors $w_i$ of the best matching unit and its neighbors are adjusted in a **Kohonen** like fashion:

$$\Delta w_{bmu} = \varepsilon_B (v - w_{bmu}) \text{ and } \Delta w_j = \varepsilon_{Nh} (v - w_j) , \qquad (2)$$

where    the    neighborhood    $Nh(j)$    of    a    unit    j    is    defined    by $Nh(j) = \{i| (C_{ji} \neq 0, 1 \leq i \leq N)\}$ .

The inner loop ends with updating the **resource** value of the best matching unit. The resource of a neuron is a local error measure attached to each neural unit. As has been pointed out, one can choose alternative update functions corresponding to different error measures. For our experiments (section 2.1 and section 3.1) we used the accumulated squared distance to the stimulus, i.e. $\Delta \tau_{bmu} = \|v - w_{bmu}\|^2$ .

The outer loop now proceeds by adding a new neural unit $r$ to the network. This unit is located in-between the unit $l$ with largest resource value and its neighbor $n$ with second largest resource value:[4]

The exact location of its centre of receptive field $w_r$ is calculated according to the ratio of the resource values $\tau_l$, $\tau_n$, and the resource values of units $n$ and $l$ are redistributed among $r$, $n$ and $l$:

$$\gamma = \tau_n / (\tau_n + \tau_l) \, , \, \Delta\tau_l = \frac{1}{2}(1-\gamma)\,\tau_l \text{ and } \Delta\tau_n = \frac{1}{2}\gamma\tau_n \qquad (3)$$

$$w_r = w_l + \gamma(w_n - w_l) \, , \, \tau_r = \Delta\tau_n + \Delta\tau_l, \, \tau_l = \tau_l - \Delta\tau_l \text{ and } \tau_n = \tau_n - \Delta\tau_n. \qquad (4)$$

This gives an estimate of the resource values if the new unit had been in the network right from the start. Finally the lateral connections are changed,

$$C_{rl} = C_{lr} = 1 \, , \, C_{rn} = C_{rn} = 1 \text{ and } C_{nl} = C_{ln} = 0, \qquad (5)$$

connecting unit $r$ to unit $l$ and disconnecting $n$ and $l$.

This heuristic guided by the lateral connection structure and the resource values promises insertion of new units at good initial positions. It is responsible for the better performance of **DCS-GCS** and **GCS** compared to algorithms which do not exploit the neighborhood relation between existing units.

The outer loop closes by decrementing the resource values of all units, $\tau_i(t+1) = \beta\tau_i(t) \, , \, 1 \le i \le N$, where $0 < \beta < 1$ is a constant. This last step just avoids overflow of the resource variables. For off-line learning, $\beta = 0$ is the natural choice.

### 2.1 Unsupervised DCS simulation results

Let us first turn to our simulation on artificial data. The training set $T$ contains 2000 examples randomly drawn from a feature manifold $M$ consisting of three squares, two of them connected by a line. The development of our unsupervised **DCS-GCS** network is depicted in Figure 1, with the initial situation of only two units shown in the upper left. Examples are represented by small dots, the centres of receptive fields by small circles and the lateral connections by lines connecting the circles. From left to right the network is examined after 0, 9 and 31 epochs of training (i.e. after insertion of 2, 11 and 33 neural units).

After 31 epochs the network has built a perfectly topology preserving map of $M$, the lateral connection structure nicely reflecting the shape of $M$: Where $M$ is 2-dimensional the lateral connection structure is 2-dimensional, and it is 1-dimensional where $M$ is 1-dimensional. Note, that a connected component analysis could recognize that the upper right square is separated from the rest of $M$. The accumulated squared distance to stimuli served as the resource.

The quantization error $E_q = \frac{1}{n}\sum_{v \in T} \|v - w_{bmu(v)}\|^2$ dropped from 100% (3 units) to 3% (33 units).

The second simulation deals with the two-spirals benchmark. Data were obtained by running the program "two-spirals" (provided by CMU) with parameters 5 (density) and 6.5 (spiral radius) resulting in a training set $T$ of 962 examples. The data represent two distinct spirals in the x-y-plane. Unsupervised **DCS-GCS** at work is shown in Figure 2, after insertion of 80, 154 and, finally, 196 units. With 196 units a perfectly topology preserving map of $M$ has emerged, and the two spirals are clearly separated. Note that the algorithm has learned the separation in a totally unsupervised manner, i.e. not using the labels of the data

---

4. Fritzke inserts new units at a slightly different location, using not the neighbor with second largest resource but the most distant neighbor.

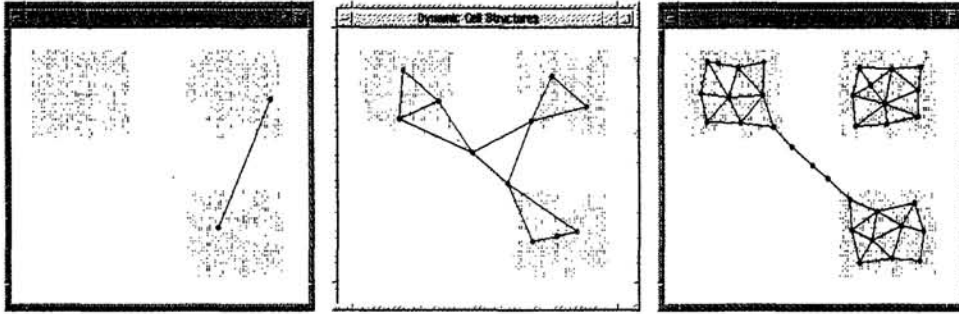

Figure 1: Unsupervised DCS-GCS on artificial data

points (which are provided by CMU for supervised learning). Again, the accumulated squared distance to stimuli served as the resource.

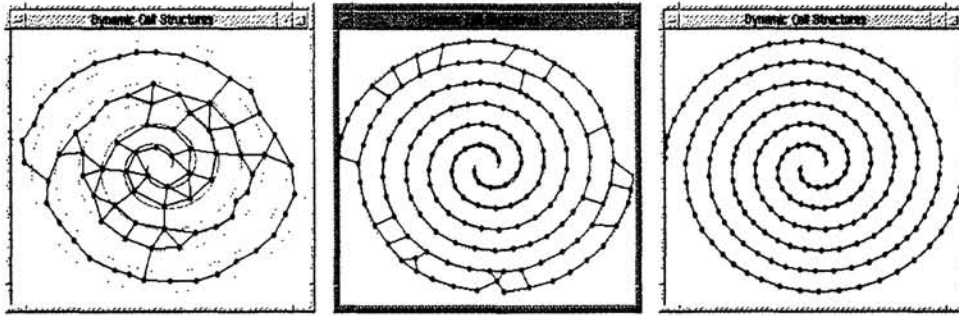

Figure 2: Unsupervised learning of two spirals

## 3 Supervised DCS-GCS

In supervised **DCS-GCS** examples consist not only of an input vector $v$ but also include an additional teaching output vector $u$.

The supervised algorithm actually does work very similar to its unsupervised version except

- when a neural unit $n_i$ is inserted an output vector $o_i$ will be attached to it with $o_i = u$.

- the output $y$ of the network is calculated as a weighted sum of the best matching unit's output vector $o_{bmu}$ and the output vectors of its neighbors $o_i$, $i \in Nh(bmu)$,

$$y = \left( \sum_{i \in \{bmu \cup Nh(bmu)\}} a_i o_i \right), \tag{6}$$

where $a_i = 1/(\sigma\|v - w_i\|^2 + 1)$ is the activation of neuron i on stimulus $v$, $\sigma, \sigma > 0$, representing the size of the receptive fields. In our simulations, the size of receptive fields have been equal for all units.

- adaption of output vectors by the delta-rule: A simple delta-rule is employed to adjust the output vectors of the best matching unit and its neighbors.

Most important, the approximation (classification) error can be used for resource updating. This leads to insertion of new units in regions where the approximation error is worst, thus promising to outperform dynamic algorithms which do not employ such a criterion for insertion. In our simulations we used the accumulated squared distance of calculated and teaching output, $\Delta\tau_{bmu} = \|y - u\|^2$.

### 3.1 Supervised DCS-GCS simulation results

We applied our supervised **DCS-GCS** algorithm to three CMU benchmarks, the supervised two-spiral problem, the speaker independent vowel recognition problem and the sonar mine/ rock separation problem.[5]

The two spirals benchmark contains 194 examples, each consisting of an input vector $v \in \mathfrak{R}^2$ and a binary label indicating to which spiral the point belongs. The spirals can not be linearly separated. The task is to train the examples until the learning system can produce the correct output for all of them and to record the time.

The decision regions learned by supervised **DCS-GCS** are depicted in Figure 3 after 110 and 135 epochs of training, where the classification error on the training set has dropped to 0%. Black indicates assignment to the fist, white assignment to the second spiral. The network and the examples are overlaid.

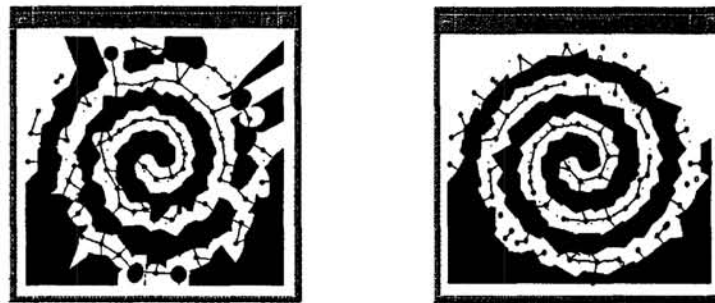

Figure 3: Supervised learning of two spirals

Results reported by others are 20000 epochs of Backprop for a MLP by Lang and Witbrok [Lang89], 10000 epochs of Cross Entropy Backprop and 1700 epochs of Cascade-Correlation by Fahlman and Lebiere [Fahlman90] and 180 epochs of **GCS** training by Fritzke [Fritzke93].

---

5. For details of simulation, parameters and additional statistics for all of the reported experiments the reader is refered to [Bruske94] which is also available via *ftp.informatik.uni-kiel.de* in directory *pub/kiel/publications/TechnicalReports/Ps.Z/* as *1994tr03.ps.Z*

The data for the speaker independent recognition of 11 vowels comprises a training set of 582 examples and a test set of 462 examples, see [Robinson89].

We obtained 65% correctly classified test samples with only 108 neural units in the **DCS-GCS** network. This is superior to conventional models (including single and multi layer perceptron, Kanerva Model, Radial Basis Functions, Gaussian Node Network, Square Node Network and Nearest Neighbor) for which figures well below 57% have been reported by Robinson. It also qualitatively compares to GCS (jumps above the 60% margin), for which Fritzke reports best classification results of 61%(158 units) up to 67% (154 units) for a 3-dim GCS. On the other hand, our best **DCS-GCS** used much fewer units. Note that **DCS-GCS** did not rely on a pre-specified connection structure (but learned it!).

Our last simulation concerns a data set used by Gorman and Sejnowski in their study of classification of sonar data, [Gorman88]. The training and the test set contain 104 examples each.

Gorman and Sejnowski report their best results of 90.4% correctly classified test examples for a standard BP network with 12 hidden units and 82.7% for a nearest neighbor classifier. Supervised **DCS-GCS** reached a peak classification rate of 95% after only 88 epochs of training.

## 4 Conclusion

We have introduced the idea of RBF networks which concurrently learn and utilize perfectly topology preserving feature maps for adaptation and interpolation. This family of ANNs, which we termed Dynamic Cell Structures, offers conceptual advantage compared to classical Kohonen type SOMs since the emerging lateral connection structure maximally preserves topology. We have discussed the DCS-GCS algorithm as an instance of DCS. Compared to its ancestor GCS of Fritzke, this algorithm elegantly avoids computational overhead for handling sophisticated data structures. If connection updates (eq.(1)) are restricted to the best matching unit and its neighbors, DCS has linear (serial) time complexity[6] and thus may also be considered as an improvement of Martinetz's Neural Gas idea[7]. Space complexity of DCS is $O(N^2)$ in general and can be shown to become linear if the feature manifold $M$ is two dimensional. The simulations on CMU-Benchmarks indicate that DCS indeed has practical relevance for classification and approximation.

Thus encouraged, we look forward to apply DCS at various sites in our active computer vision project, including image compression by dynamic vector quantization, sensorimotor maps for the oculomotor system and hand-eye coordination, cartography and associative memories. A recent application can be found in [Bruske95] where a DCS network attempts to learn a continous approximation of the Q-function in a reinforcement learning problem.

---

6. Here we refer to the serial time a DCS algorithm needs to process a single stimulus (including response calculation and adaptation).

7. The serial time complexity of the Neural Gas is $\Omega(N)$, approaching $O(N\log N)$ for $k \rightarrow N$, k the number of nearest neighbors.

## References

**[Bruske94]** J. Bruske and G. Sommer, *Dynamic Cell Structures: Radial Basis Function Networks with Perfect Topology Preservation*, Inst. f. Inf. u. Prakt. Math, CAU zu Kiel, Technical Report 9403.

**[Bruske95]** J. Bruske, I. Ahrns and G. Sommer, *Heuristic Q-Learning*, submitted to ECML 95.

**[Fahlman90]** S.E. Fahlman, C.Lebiere, *The Cascade-Correlation Learning Architecture*, Advances in Neural Information processing systems 2, Morgan Kaufman, San Mateo, pp.524-534.

**[Fahlman93]** S.E. Fahlman, *CMU Benchmark Collection for Neural Net Learning Algorithms*, Carnegie Mellon Univ., School of Computer Science, machine-readable data repository, Pittsburgh.

**[Fritzke92]** B. Fritzke, *Growing Cell Structures - a self organizing network in k dimensions*, Artificial Neural Networks 2, I.Aleksander & J.Taylor eds., North-Holland, Amsterdam, 1992.

**[Fritzke93]** B. Fritzke, *Growing Cell Structures - a self organizing network for unsupervised and supervised training*, ICSI Berkeley, Technical Report, tr-93-026.

**[Gorman88]** R.B. Gorman and T.J. Sejnowski, *Analysis of Hidden Units in a Layered Network Trained to Classify Sonar Targets*, Neural Networks, Vol.1, pp. 75-89

**[Lang89]** K.J. Lang & M.J. Witbrock, *Learning to tell two spirals apart*, Proc. of the 1988 Connectionist Models Summer School, Morgan Kaufmann, pp.52-59.

**[Martinetz92]** Thomas Martinetz, *Selbstorganisierende neuronale Netzwerke zur Bewegungssteuerung*, Dissertation, DIFKI-Verlag, 1992.

**[Martinetz93]** Thomas Martinetz, *Competitive Hebbian Learning Rule Forms Perfectly Topology Preserving Maps*, Proc. of the ICANN 93, p.426-438, 1993.

**[Martinetz94]** Thomas Martinetz and Klaus Schulten, *Topology Representing Networks*, Neural Networks, No. 7, Vol. 3, pp. 505-522, 1994.

**[Moody89]** J.Moody, C.J. Darken, *Fast Learning in Networks of Locally-Tuned Processing Units*, Neural Computation Vol.1 Num.2, Summer 1989.

**[Robinson89]** A.J. Robinson, *Dynamic Error Propagation Networks*, Cambridge Univ., Ph.D. thesis, Cambridge.

**[Villmann94]** T. Villmann and R. Der and T. Martinetz, *A Novel Approach to Measure the Topology Preservation of Feature Maps*, Proc. of the ICANN 94, 1994.
